# The Emergence of Multiple Movement Units in the Presence of Noise and Feedback Delay

**Michael Kositsky**          **Andrew G. Barto**

Department of Computer Science
University of Massachusetts
Amherst, MA 01003-4610
*{kositsky,barto}@cs.umass.edu*

## Abstract

Tangential hand velocity profiles of rapid human arm movements often appear as sequences of several bell-shaped acceleration-deceleration phases called submovements or movement units. This suggests how the nervous system might efficiently control a motor plant in the presence of noise and feedback delay. Another critical observation is that stochasticity in a motor control problem makes the optimal control policy essentially different from the optimal control policy for the deterministic case. We use a simplified dynamic model of an arm and address rapid aimed arm movements. We use reinforcement learning as a tool to approximate the optimal policy in the presence of noise and feedback delay. Using a simplified model we show that multiple submovements emerge as an optimal policy in the presence of noise and feedback delay. The optimal policy in this situation is to drive the arm's end point close to the target by one fast submovement and then apply a few slow submovements to accurately drive the arm's end point into the target region. In our simulations, the controller sometimes generates corrective submovements before the initial fast submovement is completed, much like the predictive corrections observed in a number of psychophysical experiments.

## 1 Introduction

It has been consistently observed that rapid human arm movements in both infants and adults often consist of several submovements, sometimes called "movement units" [21]. The tangential hand velocity profiles of such movements show sequences of several bell-shaped acceleration-deceleration phases, sometimes overlapping in the time domain and sometimes completely separate. Multiple movement units are observed mostly in infant reaching [5, 21] and in reaching movements by adult subjects in the face of difficult time-accuracy requirements [15]. These data provide clues about how the nervous system efficiently produces fast and accurate movements in the presence of noise and significant feedback delay. Most modeling efforts concerned with movement units have addressed only the kinematic aspects of movement, e.g., [5, 12].

We show that multiple movement units might emerge as the result of a control policy that is optimal in the face of uncertainty and feedback delay. We use a simplified dynamic model

of an arm and address rapid aimed arm movements. We use reinforcement learning as a tool to approximate the optimal policy in the presence of noise and feedback delay.

An important motivation for this research is that stochasticity inherent in the motor control problem has a significant influence on the optimal control policy [9]. We are following the preliminary work of Zelevinsky [23] who showed that multiple movement units emerge from the stochasticity of the environment combined with a feedback delay. Whereas he restricted attention to a finite-state system to which he applied dynamic programming, our model has a continuous state space and we use reinforcement learning in a simulated real-time learning framework.

## 2 The model description

The model we simulated is sketched in Figure 1. Two main parts of this model are the "RL controller" (Reinforcement Learning controller) and the "plant." The controller here represents some functionality of the central nervous system dealing with the control of reaching movements. The plant represents a simplified arm together with spinal circuitry. The controller generates the control signal, $u$, which influences how the state, $s$, of the plant changes over time. To simulate delayed feedback the state of the plant is made available to the controller after a delay period $\Delta$, so at time $t$ the controller can only observe $s\,(t - \Delta)$. To introduce stochasticity, we disturbed $u$ by adding noise to it, to produce a corrupted control $\tilde{u}$. The controller learns to move the plant state as quickly as possible into a small region about a target state $s_T$. The reward structure block in Figure 1 provides a negative unit reward when the plant's state is out of the target area of the state space, and it provides zero reward when the plant state is within the target area. The reinforcement learning controller tries to maximize the total cumulative reward for each movement. With the above mentioned reward structure, the faster the plant is driven into the target region, the less negative reward is accumulated during the movement. Thus this reward structure specifies the minimum time-to-goal criterion.

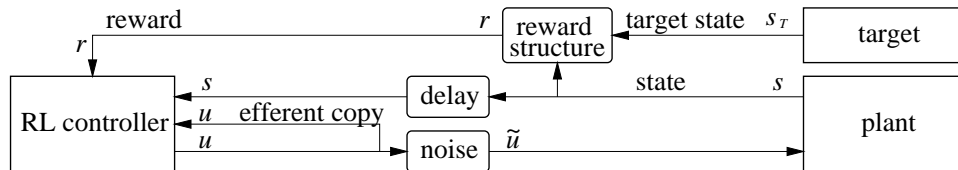

Figure 1: Sketch of the model used in our simulations. "RL controller" stands for a Reinforcement Learning controller.

### 2.1 The plant

To model arm dynamics together with the spinal reflex mechanisms we used a fractional-power damping dynamic model [22]. The simplest model that captures the most critical dynamical features is a spring-mass system with a nonlinear damping:

$$m\ddot{x} + b\dot{x}^{\frac{1}{5}} + k\,(x - u) = 0.$$

Here, $x$ is the position of the mass attached to the spring, $\dot{x}$ and $\ddot{x}$ are respectively the velocity and the acceleration of the object, $m$ is the mass of the object (the mass of the spring is assumed equal to zero), $b$ is the damping coefficient, $k$ is the stiffness coefficient, and $u$ is the control signal which determines the resting, or equilibrium, position. Later in this paper, we call $u$ activation, referring to the activation level of a muscle pair. The

Table 1: Parameter values used in the simulations.

| description | value | description | value |
|---|---|---|---|
| the basic simulation time step | 1 ms | threshold velocity radius | 0.1 cm/s |
| the feedback delay, $\Delta$ | 200 ms | standard deviation of the noise | 1 cm |
| initial position | 0 cm | value function learning rate | 0.5 |
| initial velocity | 0 cm/s | preferences learning rate | 1 |
| target position | 5 cm | discount factor, $\gamma$ | 0.9 |
| target velocity | 5 cm | bootstrapping factor, $\lambda$ | 0.9 |
| target position radius | 0.5 cm | | |

values for the mass, the damping coefficient, and the stiffness coefficient were taken from Barto et al. [3]: $m = 1\,\mathrm{kg}$, $b = 3\,\mathrm{N\,(s/m)}^{\frac{1}{5}}$, $k = 30\,\mathrm{N/m}$. These values provide movement trajectories qualitatively similar to those observed in human wrist movements [22].

The fractional-power damping in this model is motivated by both biological evidence [8, 14] and computational considerations. Controlling a system with such a concave damping function is an easier control problem than for a system with apparently simpler linear damping. Fractional-power damping creates a qualitatively novel dynamical feature called a *stiction region*, a region in the position space around the equilibrium position consisting of pseudo-stable states, where the velocity of the plant remains very close to zero. Such states are stable states for all practical purposes. For the parameter magnitudes used in our simulations, the stiction region is a region of radius 2.5 cm about the true equilibrium in the position space.

Another essential feature of the neural signal transmission can be accounted for by using a cascade of low-pass temporal filters on the activation level $u$ [16]. We used a second-order low-pass filter with the time constant of 25 ms.

## 2.2 The reinforcement learning controller

We used the version of the actor-critic algorithm described by Sutton and Barto [20]. A possible model of how an actor-critic architecture might be implemented in the nervous system was suggested by Barto [2] and Houk et al. [10]. We implemented the actor-critic algorithm for a continuous state space and a finite set of actions, i.e., activation level magnitudes $u$ evenly spaced every 1 cm between 0 cm and 10 cm. To represent functions defined over the continuous state space we used a CMAC representation [1] with 10 tilings, each tiling spans all three dimensions of the state space and has 10 tiles per dimension. The tilings have random offsets drawn from the uniform distribution. Learning is done in episodes. At the beginning of each episode the plant is at a fixed initial state, and the episode is complete when the plant reaches the target region of the state space. Table 1 shows the parameter values used in the simulations. Refer to ref. [20] for algorithm details.

## 2.3 Clocking the control signal

For the controller to have sufficient information about the current state of the plant, the controller internal representation of the state should be augmented by a vector of all the actions selected during the last delay period. To keep the dimension of the state space at a feasible level, we restrict the set of available policies and make the controller select a new activation level, $u$, in a clocked manner at time intervals equal to the delay period. One step of the reinforcement learning controller is performed once a delay period, which corresponds to many steps of the underlying plant simulation. To simulate a stochastic plant we added Gaussian noise to $u$. A new Gaussian disturbance was drawn every time a

new activation level was selected.

Apart from the computational motivation, there is evidence of intermittent motor control by human subjects [13]. In our simulations we use an oversimplified special kind of intermittent control with a piecewise constant control signal whose magnitude changes at equal time intervals, but this is done for the sake of acceleration of the simulations and overall clarity. Intermittent control does not necessarily assume this particular kind of the control signal; the most important feature is that control segments are selected at particular points in time, and each control segment determines the control signal for an extended time interval. The time interval until selection of the next control segment can itself be one of the parameters [11].

## 3  Results

The model learned to move the mass quickly and accurately to the target in approximately 1,000 episodes. Figure 2 shows the corresponding learning curve. Figure 3 shows a typical movement accomplished by the controller after learning. The movement shown in Figure 3 has two acceleration-deceleration phases called movement units or submovements.

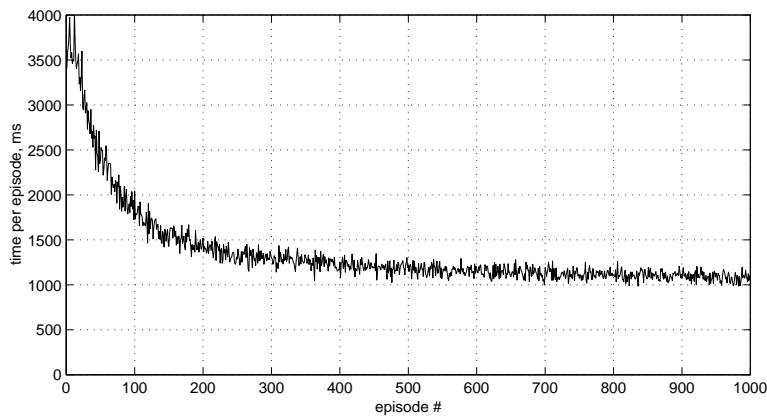

Figure 2: The learning curve averaged over 100 trials. The performance is measured in time-per-episode.

Corrective submovements may occur before the plant reaches zero velocity. The controller generates this corrective submovement "on the fly," i.e., before the initial fast submovement is completed. Figure 4 shows a sample movement accomplished by the controller after learning where such overlapping submovements occur. This can be seen clearly in panel (b) of Figure 4 where the velocity profile of the movement is shown. Each of the submovements appears as a bell-shaped unit in the tangential velocity plot.

Sometimes the controller accomplishes a movement with a single smooth submovement. A sample of such a movement is shown in Figure 5.

## 4  Discussion

The model learns to produce movements that are fast and accurate in the presence of noise and delayed sensory feedback. Most of the movements consist of several submovements. The first submovement is always fast and covers most of the distance from the initial po-

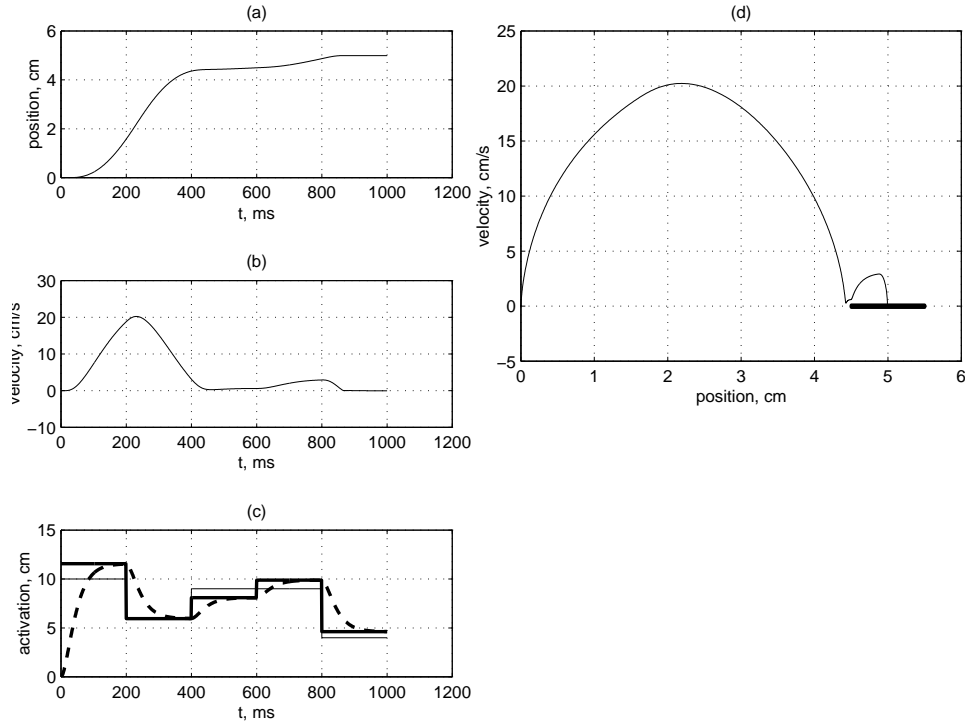

Figure 3: A sample movement accomplished by the controller after learning. Panels (a) and (b) show the position and velocity time course respectively. Panel (c) shows the activation time courses. The thin solid line shows the activation $u$ selected by the controller. The thick solid line shows the disturbed activation $\tilde{u}$ which is sent as the control signal to the plant. The dashed line shows the activation after the temporal filtering is applied. Panel (d) shows the phase trajectory of the movement. The thick bar at the lower-right corner is the target region.

sition to the target. All of the subsequent submovements are much slower and cover much shorter segments in the position space.

This feature stands in good agreement with the dual control model [12, 17], where the initial part of a movement is conducted in a ballistic manner, and the final part is conducted under closed-loop control. Some evidence for this kind of dual control strategy comes from experiments in which subjects were given visual feedback only during the initial stage of movement. Subjects did not show significant improvement under these conditions compared to trials in which they were deprived of visual feedback during the entire movement [4, 6]. In another set of experiments, proprioceptive feedback was altered by stimulations of muscle tendons. Movement accuracy decreased only when the stimulation was applied at the final stages of movement [18]. Note, however, that the dual control strategy though is not explicitly designed into our model, but naturally emerges from the existing constraints and conditions.

The reinforcement learning controller is encouraged by the reward structure to accomplish each movement as quickly as possible. On the other hand, it faces high uncertainty in the plant behavior. In states with low velocities the information available to the controller determines the actual state of the plant quite accurately as opposed to states with high

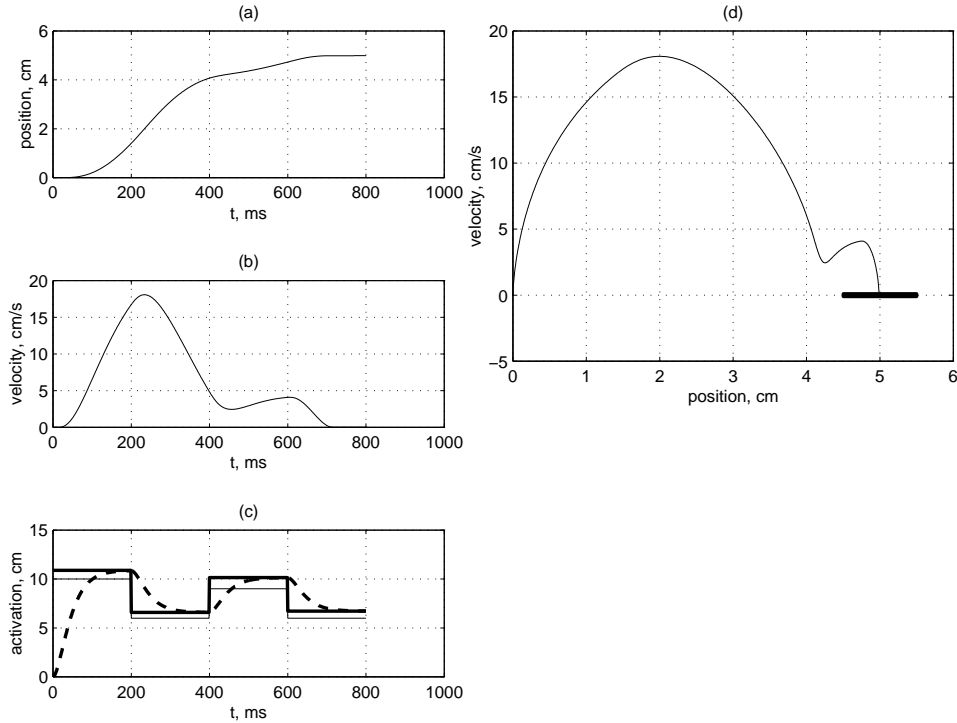

Figure 4: A sample movement accomplished by the controller after learning with a well expressed predictive correction.

velocities. If the controller were to adopt a policy in which it attempts to directly hit the target in one fast submovement, then very often it would miss the target and spend long additional time to accomplish the task. The optimal policy in this situation is to move the arm close to the target by one fast submovement and then apply a few slow submovements to accurately move arm into the target region.

The model learns to produce control sequences consisting of pairs of high activation steps followed by low activation steps. This feature stands in good agreement with pulse-step models of motor control [7, 19]. Each of the pulse-step combinations produces a submovement characterized by a bell-shaped unit in the velocity profile.

In biological motor control corrective submovements are observed very consistently, including both the overlapping and separate submovements. In the case of overlapping submovements, the corrective movement is called a predictive correction. Multiple submovements are observed mostly in infant reaching [5]. Adults perform routine everyday reaching movements ordinarily with a single smooth submovement, but in case of tight time constraints or accuracy requirements they revert to multiple submovements [15]. The suggested model sometimes accomplishes movements with a single smooth submovement (see Figure 5), but in most cases it produces multiple submovements much like an infant or an adult subject trying to move quickly and accurately.

The suggested model is also consistent with theories of basal ganglia information processing for motor control [10]. Some of these theories suggest that dopamine neurons in the basal ganglia carry information similar to the secondary reinforcement (or temporal difference) in the actor-critic controller, i.e., information about how the expected perfor-

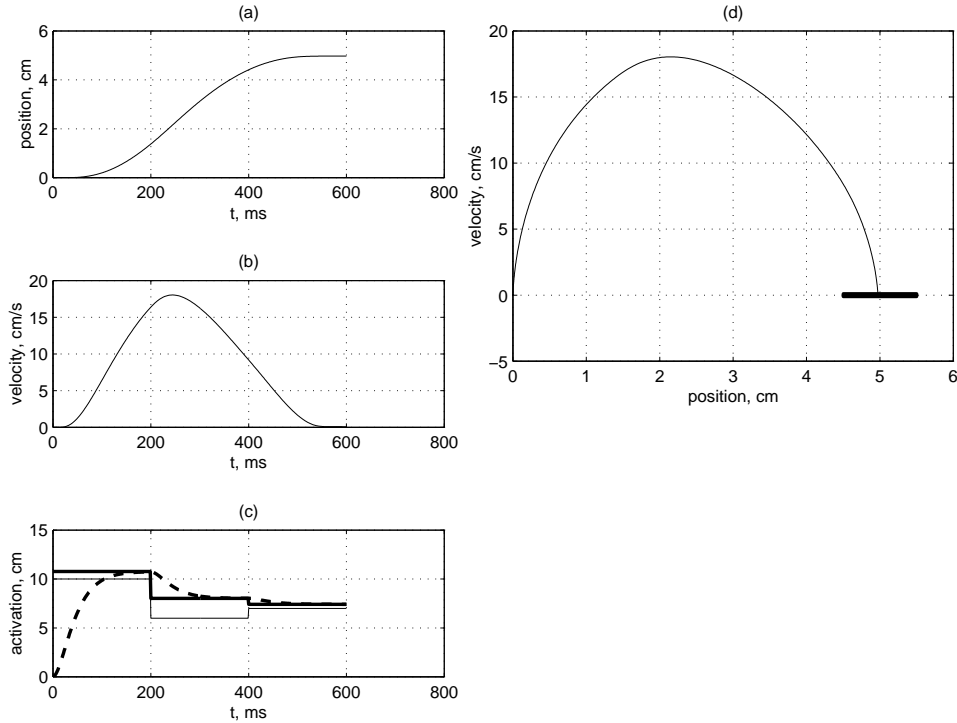

Figure 5: A sample movement accomplished by the controller after learning with a single smooth submovement.

mance (time-to-target) changes over time during a movement. A possible use of this kind of information is for initiating corrective submovements before the current movement is completed. This kind of behavior is exhibited by our model (Figure 4).

### Acknowledgments

This work was supported by NIH Grant MH 48185–09. We thank Andrew H. Fagg and Michael T. Rosenstein for helpful comments.

## References

[1] J. S. Albus. A new approach to manipulator control: the cerebellar model articulation controller (CMAC). *Journal of Dynamics, Systems, Measurement and Control*, 97:220–227, 1975.

[2] A. G. Barto. Adaptive critics and the basal ganglia. In J. C. Houk, J. L. Davis, and D. G. Beiser, editors, *Models of Information Processing in the Basal Ganglia*, pages 215–232. MIT Press, Cambridge, MA, 1995.

[3] A. G. Barto, A. H. Fagg, N. Sitkoff, and J. C. Houk. A cerebellar model of timing and prediction in the control of reaching. *Neural Computation*, 11:565–594, 1999.

[4] D. Beaubaton and L. Hay. Contribution of visual information to feedforward and feedback processes in rapid pointing movements. *Human Movement Science*, 5:19–34, 1986.

[5] N. E. Berthier. Learning to reach: a mathematical model. *Developmental Psychology*, 32:811–832, 1996.

[6] L. G. Carlton. Processing of visual feedback information for movement control. *Journal of Experimental Psychology: Human Perception and Performance*, 7:1019–1030, 1981.

[7] C. Ghez. Contributions of central programs to rapid limb movement in the cat. In H. Asanuma and V. J. Wilson, editors, *Integration in the Nervous System*, pages 305–320. Igaku-Shoin, Tokyo, 1979.

[8] C. C. A. M. Gielen and J. C. Houk. A model of the motor servo: incorporating nonlinear spindle receptor and muscle mechanical properties. *Biological Cybernetics*, 57:217–231, 1987.

[9] C. M. Harris and D. M. Wolpert. Signal-dependent noise determines motor planning. *Nature*, 394:780–784, 1998.

[10] J. C. Houk, J. L. Adams, and A. G. Barto. A model of how the basal ganglia generates and uses neural signals that predict reinforcement. In J. C. Houk, J. L. Davis, and D. G. Beiser, editors, *Models of Information Processing in the Basal Ganglia*, pages 249–270. MIT Press, Cambridge, MA, 1995.

[11] M. Kositsky. *Motor Learning and Skill Acquisition by Sequences of Elementary Actions*. PhD thesis, The Weizmann Institute of Science, Israel, October 1998.

[12] D. E. Meyer, S. Kornblum, R. A. Abrams, C. E. Wright, and J. E. K. Smith. Optimality in human motor performance: ideal control of rapid aimed movements. *Psychological Review*, 95(3):340–370, 1988.

[13] R. C. Miall, D. J. Weir, and J. F. Stein. Intermittency in human manual tracking tasks. *Journal of Motor Behavior*, 25:53–63, 1993.

[14] L. E. Miller. Reflex stiffness of the human wrist. Master's thesis, Department of Physiology, Northwestern University, Evanston, IL, 1984.

[15] K. E. Novak, L. E. Miller, and J. C. Houk. Kinematic properties of rapid hand movements in a knob turning task. *Experimental Brain Research*, 132:419–433, 2000.

[16] L. D. Partridge. Integration in the central nervous system. In J. H. U. Brown and S. S. Gann, editors, *Engineering Principles in physiology*, pages 47–98. Academic Press, New York, 1973.

[17] R. Plamondon and A. M. Alimi. Speed/accuracy trade-offs in target-directed movements. *Behavioral and Brain Science*, 20:279–349, 1997.

[18] C. Redon, L. Hay, and J.-L. Velay. Proprioceptive control of goal directed movements in man studied by means of vibratory muscle tendon stimulation. *Journal of Motor Behavior*, 23:101–108, 1991.

[19] D. A. Robinson. Oculomotor control signals. In G. Lennerstrand and P. B. y Rita, editors, *Basic Mechanisms of Ocular Mobility and Their Clinical Implications*, pages 337–374. Pergamon Press, Oxford, 1975.

[20] R. S. Sutton and A. G. Barto. *Reinforcement Learning: An Introduction*. MIT Press, Cambridge, MA, 1998.

[21] C. von Hofsten. Structuring of early reaching movements: A longitudinal study. *Journal of Motor Behavior*, 23:280–292, 1991.

[22] C. H. Wu, J. C. Houk, K. Y. Young, and L. E. Miller. Nonlinear damping of limb motion. In J. M. Winters and S. L.-Y. Woo, editors, *Multiple Muscle Systems: Biomechanics and Movement Organization*, pages 214–235. Springer-Verlag, New York, 1990.

[23] L. Zelevinsky. Does time-optimal control of a stochastic system with sensory delay produce movement units? Master's thesis, University of Massachusetts, Amherst, 1998.
